# Adaptation and Unsupervised Learning

Peter Dayan    Maneesh Sahani    Grégoire Deback
Gatsby Computational Neuroscience Unit
17 Queen Square, London, England, WC1N 3AR.
{dayan, maneesh}@gatsby.ucl.ac.uk, gdeback@ens-lyon.fr

## Abstract

Adaptation is a ubiquitous neural and psychological phenomenon, with a wealth of instantiations and implications. Although a basic form of plasticity, it has, bar some notable exceptions, attracted computational theory of only one main variety. In this paper, we study adaptation from the perspective of factor analysis, a paradigmatic technique of unsupervised learning. We use factor analysis to re-interpret a standard view of adaptation, and apply our new model to some recent data on adaptation in the domain of face discrimination.

## 1  Introduction

Adaptation is one of the first facts with which neophyte neuroscientists and psychologists are presented. Essentially all sensory and central systems show adaptation at a wide variety of temporal scales, and to a wide variety of aspects of their informational milieu. Adaptation is a product (or possibly by-product) of many neural mechanisms, from short-term synaptic facilitation and depression,[1] and spike-rate adaptation,[28] through synaptic remodeling[27] and way beyond. Adaptation has been described as the psychophysicist's electrode, since it can be used as a sensitive method for revealing underlying processing mechanisms; thus it is both phenomenon and tool of the utmost importance.

That adaptation is so pervasive makes it most unlikely that a single theoretical framework will be able to provide a compelling treatment. Nevertheless, adaptation should be just as much a tool for theorists interested in modeling neural statistical learning as for psychophysicists interested in neural processing. Put abstractly, adaptation involves short or long term changes to aspects of the statistics of the environment experienced by a system. Thus, accounts of neural plasticity driven by such statistics, even if originally conceived as accounts of developmental (or perhaps representational) plasticity,[19] are automatically candidate models for the course and function of adaptation. Conversely, thoughts about adaptation lay at the heart of the earliest suggestions that redundancy reduction and information maximization should play a central role in models of cortical unsupervised learning.[4–6, 8, 23]

Redundancy reduction theories of adaptation reached their apogee in the work of Linsker,[26] Atick, Li & colleagues[2, 3, 25] and van Hateren.[40] Their mathematical framework (see section 2) is that of maximizing information transmission subject to various sources of noise and limitations on the strength of key signals. Noise plays the critical roles of rendering some signals essentially undetectable, and providing a confusing background against which other signals should be amplified. Adaptation, by affecting noise levels and informational content (notably probabilistic priors), leads to altered stimulus processing. Early work concentrated on the effects of sensory noise on visual receptive fields; more recent studies[41] have used the same framework to study stimulus specific adaptation.

Redundancy reduction is one major conceptual plank in the modern theory of unsupervised learning. However, there are various other important complementary ideas, notably gen-

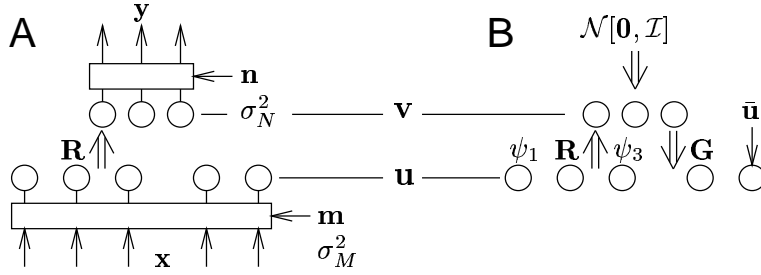

**Figure 1**: A) Redundancy reduction model. $\mathbf{u}$ is the explicit input, combining signal $\mathbf{x}$ and noise $\mathbf{m}$; $\mathbf{v}$ is the explicit output, to be corrupted by noise $\mathbf{n}$ to give $\mathbf{y}$. We seek the filter $\mathbf{R}$ that minimizes redundancy subject to a power constraint. B) Factor analysis model. Now $\mathbf{v}$, with a white, Gaussian, prior, captures latent structure underlying the covariance $\mathcal{U}$ of $\mathbf{u}$. The empirical mean is $\bar{\mathbf{u}}$; the uniquenesses $\psi_i$ capture unmodeled variance and additional noise such as $\sigma_M^2$. Generative $\mathbf{G}$ and recognition $\mathbf{R}$ weights parameterize statistical inverses.

erative models.[19] Here, we consider adaptation from the perspective of factor analysis,[15] which is one of the most fundamental forms of generative model. After describing the factor analysis model and its relationship with redundancy reduction models of adaptation in section 3, section 4 studies loci of adaptation in one version of this model. As examples, we consider adaptation of early visual receptive fields to light levels,[38] orientation detection to a persistent bias (the tilt aftereffect),[9,16] and a recent report of adaptation of face discrimination to morphed anti-faces.[24]

## 2 Information Maximization

Figure 1,[3] shows a linear model of, for concreteness, retinal processing. Here, $m$-dimensional photoreceptor input $\mathbf{u} = \mathbf{x} + \mathbf{m}$, which is the sum of a signal $\mathbf{x}$ and detector noise $\mathbf{m}$, is filtered by a retinal matrix to produce an $n$-dimensional output $\mathbf{v} = \mathbf{R}\mathbf{u}$ for communication down the optic nerve $\mathbf{y} = \mathbf{v} + \mathbf{n}$, against a background of additional noise $\mathbf{n}$. We assume that the signal is Gaussian, with mean $\mathbf{0}$ and covariance $\mathcal{X}$, and the noise terms are white and Gaussian, with mean $\mathbf{0}$ and covariances $\sigma_M^2 \mathcal{I}$ and $\sigma_N^2 \mathcal{I}$, respectively; all are mutually independent. The input may be higher dimensional than the output, *ie* $m > n$, as is true of the retina. Here, the signal is translation invariant, *ie* $\mathcal{X}$ is a circulant matrix[11] with $\mathcal{X}_{ab} \equiv X(a - b)$. This means that the eigenvectors of $\mathcal{X}$ are (discrete) sine and cosines, with eigenvalues coming from the Fourier series for $X$, whose terms we will write as $x^1 \geq x^2 \geq \ldots > 0$ (they are non-negative since $\mathcal{X}$ is a covariance matrix; we assume for simplicity that they are strictly positive).

Given no input noise ($\sigma_M^2 = 0$), the mutual information between $\mathbf{u} = \mathbf{x}$ and $\mathbf{y}$ is

$$\mathcal{I}[\mathbf{x}, \mathbf{y}] = \mathcal{H}[\mathbf{y}] - \mathcal{H}[\mathbf{y}|\mathbf{x}] = (\ln |\mathbf{R}\mathcal{X}\mathbf{R}^T + \sigma_N^2 \mathcal{I}| - \ln |\sigma_N^2 \mathcal{I}|)/2 \tag{1}$$

where $\mathcal{H}$ is the entropy function (which, for a Gaussian distribution, is proportional to the ln determinant of its covariance matrix). We consider maximizing this with respect to $\mathbf{R}$, a calculation which only makes sense in the face of a constraint, such as on the average power $\langle |\mathbf{v}|^2 \rangle = \text{tr} \left( \mathbf{R}\mathcal{X}\mathbf{R}^T \right)$. It is a conventional result in principal components analysis[12,20] that the solution to this constrained maximization problem involves whitening, *ie* making

$$\mathbf{R} = \mathbf{A}\mathbf{D}\mathbf{E}^n \quad \text{with} \quad \mathbf{D} \propto \text{diag} \left\{ \frac{1}{\sqrt{x_1}}, \frac{1}{\sqrt{x_2}}, \ldots, \frac{1}{\sqrt{x_n}} \right\} \tag{2}$$

where $\mathbf{A}$ is an arbitrary $n$-dimensional rotation matrix with $\mathbf{A}\mathbf{A}^T = \mathcal{I}$, $\mathbf{D}$ is the $n \times n$ diagonal matrix with the given form, and $\mathbf{E}^n$ is an $n \times m$ matrix whose rows are the first $n$ (transposed) eigenvectors of $\mathcal{X}$. This choice makes $\mathbf{R}\mathcal{X}\mathbf{R}^T \propto \mathcal{I}$, and effectively amplifies weak input channels (*ie* those with small $x_k$) so as fully to utilize all the output channels.

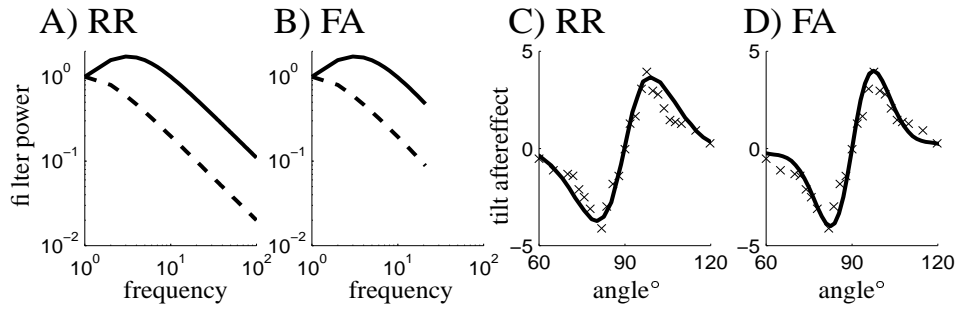

**Figure 2**: Simple adaptation. A;B) Filter power as a function of spatial frequency for the redundancy reduction (A: RR) and factor analysis (B: FA) solutions for the case of translation invariance, for low (solid: $\sigma_M^2 = 0.1$) and high (dashed $\sigma_M^2 = 1$) input noise and $x_k = 1/k^2$. Even though the optimal FA solution does have exactly identical uniquenesses, the difference is too small to figure. In (B), 40 factors were found for 256 inputs. C) Data[9] (crosses) and RR solution[41] (solid) for the tilt aftereffect. D) Data (crosses) and linear approximate FA solution (solid). For FA, angle estimation is based on the linear output of the single factor; linearity breaks down for $|\theta - 90°| > 30°$. Adaptation was based on reducing the uniquenesses $\psi_i$ for units activated by the adapting stimulus (fitting the width and strength of this adaptation to the data).

In the face of input noise, whitening is dangerous for those channels for which $\sigma_M^2 \gtrsim x_k$, since noise rather than signal would be amplified by the $1/\sqrt{x_k}$. One heuristic is to pre-filter $\mathbf{u}$ using an $m$-dimensional matrix $\mathbf{F}$ such that $\mathbf{Fu}$ is the prediction of $\mathbf{x}$ that minimizes the average error $\langle |\mathbf{Fu} - \mathbf{x}|^2 \rangle$, and then apply the $\mathbf{R}$ of equation 2.[14] Another conventional result[12] is that $\mathbf{F}$ has a similar form to $\mathbf{R}$, except that $\mathbf{A} = [\mathbf{E}^m]^T$, and the diagonal entries of the equivalent of $\mathbf{D}$ are $x_k/(x_k + \sigma_M^2)$. This makes the full (approximate) filter

$$\mathbf{R} = \mathbf{ADE}^n \quad \text{with} \quad \mathbf{D} \propto \text{diag}\left\{ \frac{\sqrt{x_1}}{x_1 + \sigma_M^2}, \frac{\sqrt{x_2}}{x_2 + \sigma_M^2}, \ldots, \frac{\sqrt{x_n}}{x_n + \sigma_M^2} \right\} \tag{3}$$

Figure 2A shows the most interesting aspect of this filter in the case that $x_k = 1/k^2$, inspired by the statistics of natural scenes,[36] for which $k$ might be either a temporal or spatial frequency. The solid curve shows the diagonal components of $\mathbf{D}$ for small input noise. This filter is a band-pass filter. Intermediate frequencies with input power well above the noise level $\sigma_M^2$ are comparatively amplified against the output noise $\mathbf{n}$, On the other hand, the dashed line shows the same components for high input noise. This filter is a low-pass filter, as only those few components with sufficient input power are significantly transmitted. The filter in equation 3 is based on a heuristic argument. An exact argument[2,3] leads to a slightly more complicated form for the optimal filter, in which, depending on the power constraint and the exact value of $\sigma_M^2$, there is a sharp cut-off in which some frequencies are not transmitted at all. However, the main pattern of dependence on $\sigma_M^2$ is the same as in figure 2A; the differences lie well outside the realm of experimental test.

Figure 2A shows a powerful form of adaptation.[3] High relative input noise arises in cases of low illumination; low noise in cases of high illumination. The whole filtering characteristics of the retina should change, from low-pass (smoothing in time or space) to band-pass (differentiation in space or time) filtering. There is evidence that this indeed happens, with dendritic remodeling happening over times of the order of minutes.[42]

Wainwright[41] (see also[10]) suggested an account along exactly these lines for more stimulus-specific forms of adaptation such as the tilt aftereffect shown in figure 2C. Here (conceptually), subjects are presented with a vertical grating ($\theta = 90°$) for an adapting period of a few seconds, and then are asked, by one of a number of means, to assess the orientation of test gratings. The crosses in figure 2C shows the *error* in their estimates; the adapting orientation appears to repel nearby angles, so that true values of $\theta$ near $90°$ are reported

as being further away. Wainwright modeled this in the light of a neural population code for representing orientation and a filter related to that of equation 3. He suggested that during adaptation, the signal associated with $\theta = 90°$ is temporarily increased. Thus, as in the solid line of figure 2A, the transmission through the adapted filter of this signal should be temporarily *reduced.* If the recipient structures that use the equivalent of $\mathbf{v}$ to calculate the orientation of a test grating are unaware of this adaptation, then, as in the solid line of figure 2C, an estimation error like that shown by the subjects will result.

## 3 Factor Analysis and Adaptation

We sought to understand the adaptation of equation 3 and figure 2A in a factor analysis model. Factor analysis[15] is one of the simplest probabilistic generative schemes used to model the unsupervised learning of cortical representations, and underlies many more sophisticated approaches. The case of uniform input noise $\sigma_M^2$ is particularly interesting, because it is central to the relationship between factor analysis and principal components analysis.[20,34,39]

Figure 1B shows the elements of a factor analysis model (see Dayan & Abbott[12] for a relevant tutorial introduction). The (so-called) visible variable $\mathbf{u}$ is generated from the latent variable $\mathbf{v}$ according to the two-step

$$p[\mathbf{v}] \sim \mathcal{N}[\mathbf{0}, \mathcal{I}] \quad p[\mathbf{u}|\mathbf{v}] \sim \mathcal{N}[\mathbf{G}\mathbf{v} + \bar{\mathbf{u}}, \boldsymbol{\Psi}] \text{ with } \boldsymbol{\Psi} = \text{diag}(\psi_1, \dots, \psi_m) \tag{4}$$

where $\mathcal{N}[\boldsymbol{\omega}, \Omega]$ is a multi-variate Gaussian distribution with mean $\boldsymbol{\omega}$ and covariance matrix $\Omega$, $\mathbf{G}$ is a set of top-down generative weights, $\bar{\mathbf{u}}$ is the mean of $\mathbf{u}$, and $\boldsymbol{\Psi}$ a diagonal matrix of *uniquenesses,* which are the variances of the residuals of $\mathbf{u}$ that are not represented in the covariances associated with $\mathbf{v}$. Marginalizing out $\mathbf{v}$, equation 4 specifies a Gaussian distribution for $\mathbf{u} \sim \mathcal{N}[\bar{\mathbf{u}}, \mathbf{G}\mathbf{G}^T + \boldsymbol{\Psi}]$, and, indeed, the maximum likelihood values for the parameters given some input data $\mathbf{u}$ are to set $\bar{\mathbf{u}}$ to the empirical mean of the $\mathbf{u}$ that are presented, and to set $\mathbf{G}$ and $\boldsymbol{\Psi}$ by maximizing the likelihood of the empirical covariance matrix $\mathcal{U}$ of the $\mathbf{u}$ under a Wishart distribution with mean $\mathbf{G}\mathbf{G}^T + \boldsymbol{\Psi}$. Note that $\mathbf{G}$ is only determined up to an $n \times n$ rotation matrix $\mathbf{A}$, since $(\mathbf{G}\mathbf{A})(\mathbf{G}\mathbf{A})^T = \mathbf{G}\mathbf{G}^T$.

The generative or synthetic model of equation 4 shows how $\mathbf{v}$ determines $\mathbf{u}$. In most instances of unsupervised learning, the focus is on the recognition or analysis model,[30] which maps a presented input $\mathbf{u}$ into the values of the latent variable $\mathbf{v}$ which might have generated it, and thereby form its possible internal representations. The recognition model is the statistical inverse of the generative model and specifies the Gaussian distribution:

$$p[\mathbf{v}|\mathbf{u}] \sim \mathcal{N}[\mathbf{R}(\mathbf{u} - \bar{\mathbf{u}}), \Sigma] \text{ with } \Sigma = (\mathcal{I} + \mathbf{G}^T\boldsymbol{\Psi}^{-1}\mathbf{G})^{-1} \quad \mathbf{R} = \Sigma\mathbf{G}^T\boldsymbol{\Psi}^{-1} . \tag{5}$$

The mean value of $\mathbf{v}$ can be derived from the differential equation[31,32]

$$\dot{\mathbf{v}} = -\mathbf{v} + \mathbf{S}^T\boldsymbol{\Psi}^{-1}(\mathbf{u} - \bar{\mathbf{u}} - \mathbf{G}\mathbf{v}) \tag{6}$$

in which $\mathbf{u} - \bar{\mathbf{u}} - \mathbf{G}\mathbf{v}$, which is the prediction error for $\mathbf{u}$ based on the current value of $\mathbf{v}$, is downweighted according to the inverse uniquenesses $\boldsymbol{\Psi}^{-1}$, mapped through bottom-up weights $\mathbf{S}$ and left to compete against the contribution of the prior for $\mathbf{v}$ (which is responsible for the $-\mathbf{v}$ term in equation 6). For this scheme to give the right answer, the bottom-up weights should be the transpose of the top-down weights $\mathbf{S} = \mathbf{G}^T$. However, we later consider forms of adaptation that weaken this dependency.

In general, factor analysis and principal components analysis lead to different results. Indeed, although the latter is performed by an eigendecomposition of the covariance matrix of the inputs, the former requires execution of one of a variety of iterative procedures on the same covariance matrix.[21,22,35] However, if the uniquenesses are forced to be equal, *ie* $\psi_i = \psi, \forall i$, then these procedures are almost the same.[34,39] In this case, assuming that

$\bar{\mathbf{u}} = \mathbf{0},$

$$\mathbf{G}^T = \mathbf{A}\mathbf{D}'\mathbf{E}^n \text{ with } \mathbf{D}' = \text{diag}\left\{\sqrt{(u_1 - \psi)}, \sqrt{(u_2 - \psi)}, \dots, \sqrt{(u_n - \psi)}\right\} \tag{7}$$

$$\psi = \left(\sum_{i=n+1}^{m} u_i\right)/(m - n) \tag{8}$$

with the same conventions as in equation 2, except that $u_i$ are the (ordered) eigenvalues of the covariance matrix $\mathcal{U}$ of the visible variables $\mathbf{u}$ rather than explicitly of the signal. Here $\psi$ has the natural interpretation of being the average power of the unexplained components. Applying this in equation 5:

$$\mathbf{R} = \mathbf{A}\mathbf{D}\mathbf{E}^n \text{ with } \mathbf{D} = \text{diag}\left\{\frac{\sqrt{(u_1 - \psi)}}{u_1}, \frac{\sqrt{(u_2 - \psi)}}{u_2}, \dots, \frac{\sqrt{(u_n - \psi)}}{u_n}\right\}. \tag{9}$$

If $\mathbf{u}$ really comes from a signal and noise model as in figure 1, then $u_i = x_i + \sigma_M^2$, and $\psi = \psi_x + \sigma_M^2$, where $\psi_x$ is the residual uniqueness of equation 8 in the case that $\sigma_M^2 = 0$. This makes the recognition weights of equation 9

$$\mathbf{R} = \mathbf{A}\mathbf{D}\mathbf{E}^n \text{ with } \mathbf{D} = \text{diag}\left\{\frac{\sqrt{(x_1 - \psi_x)}}{x_1 + \sigma_M^2}, \frac{\sqrt{(x_2 - \psi_x)}}{x_2 + \sigma_M^2}, \dots, \frac{\sqrt{(x_n - \psi_x)}}{x_n + \sigma_M^2}\right\}. \tag{10}$$

The similarity between this and the approximate redundancy reduction expression of equation 3 is evident. Just like that filter, adaptation to high and low light levels (high and low signal/noise ratios), leads to a transition from bandpass to lowpass filtering in $\mathbf{R}$. The filter of equation 3 was heuristic; this is exact. Also, there is no power constraint imposed; rather something similar derives from the generative model's prior over the latent variables $\mathbf{v}$.

This analysis is particularly well suited to the standard treatment of redundancy reduction case of figure 2A, since adding independent noise of the same strength $\sigma_M^2$ to each of the input variables can automatically be captured by adding $\sigma_M^2$ to the common uniqueness $\psi$. However, even though the signal $\mathbf{x}$ is translation invariant in this case, it need not be that the maximum likelihood factor analysis solution has the property that $\Psi$ is proportional to $\mathcal{I}$. However, it is to a close approximation, and figure 2B shows that the strength of the principal components of $\mathcal{X}$ in the maximum likelihood $\mathbf{R}$ (evaluated as in the figure caption) shows the same structure of adaptation as in the probabilistic principal components solution, as a function of $\sigma_M^2$.

Figure 2D shows a version of the tilt illusion coming from a factor analysis model given population coded input (with Gaussian tuning curves with an orientation bandwidth of $20°$) and a single factor. It is impossible to perform the full non-linear computation of extracting an angle from the population activity $\mathbf{u}$ in a single linear operation $\mathbf{R}(\mathbf{u} - \bar{\mathbf{u}})$. However, in a regime in which a linear approximation holds, the one factor can represent the systematic covariation in the activity of the population coming from the single dimension of angular variation in the input. For instance, around $\theta = 90°$, this regime comprises roughly $\theta \in [60°, 120°]$. A close match in this model to Wainwright's[41] suggestion is that the uniquenesses $\psi_i$ for the input units (around $\theta = 90°$) that are reliably activated by an adapting stimulus should be *decreased,* as if the single factor would predict a greater proportion of the variability in the activation of those units.* This makes $\mathbf{R}$ of equation 5 more sensitive to small variations in $\mathbf{u}$ away from $\theta = 90°$, and so leads to a tilt aftereffect as an estimation bias. Figure 2D shows the magnitude of this effect in the linear regime. This is a rough match for the data in figure 2C. Our model also shows the same effect as Wainwright's[41] in orientation discrimination, boosting sensitivity near the adapted $\theta$ and reducing it around half a tuning width away.[33]

## 4   Adaptation for Faces

Another, and even simpler, route to adaptation is changing $\bar{\mathbf{u}}$ towards the mean of the recently presented (*ie* the adapting) stimuli. We use this to model a recently reported effect of adaptation on face discrimination.[24]

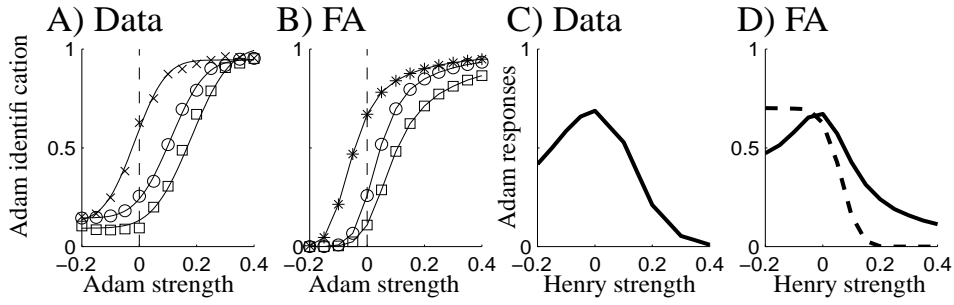

**Figure 3**: Face discrimination. Here, Adam and Henry are used for concreteness; all results are averages over all faces, and, for FA, 5000 random draws. A) Experimental[24] mean propensity to report Adam as a function of the strength of Adam in the input for no adaptation ('o'); adaptation to anti-Adam ('x'); and adaptation to anti-Henry ('□'). The curves are cumulative normal fits. B) Mean propensity in the factor analysis model for the same outcomes. The model, like some subjects, is more extreme than the mean of the subjects, particularly for test anti-faces. C;D) Experimental and model proportion of reports of Adam when adaptation was to anti-Adam; but various strengths of Henry are presented. The model captures the *decrease* in Adam given presentation of anti-Henry through a normalization pool (solid); although it does not decrease to quite the same extent as the data. Just reporting the face with the largest $v_i$ (dashed) shows no decrease in reporting Adam given presentation of anti-Henry. Here $\sigma = 0.15, \gamma = 2, \mu = 0.09$ (except for the dashed line in D, for which $\mu = 0.048$ to match the peak of the solid curve).

Leopold and his colleagues[24] studied adaptation in the complex stimulus domain of faces. Their experiment involved four target faces (associated with names 'Adam', 'Henry', 'Jim', 'John') which were previously unfamiliar to subjects, together with morphed versions of these faces lying on 'lines' going through the target faces and the average of all four faces. These interpolations were made visually sensible using a dense correspondence map between the faces. The task for the subjects was always to identify which of the four faces was presented; this is obviously impossible at the average face, but becomes progressively easier as the average face is morphed progressively further (by an amount called its strength) towards one of the target faces. The circles in figure 3A show the mean performance of the subjects in choosing the correct face as a function of its strength; performance is essentially perfect 40% of the way to the target face.

A *negative* strength version of one of the target faces (*eg* anti-Adam) was then shown to the subjects for 5 seconds before one of the positive strength faces was shown as a test. The other two lines in figure 3A show that the effect of adaptation is to boost the effective strength of the given face (Adam), since (crosses) the subjects were much readier to report Adam, even for the average face (which contains no identity information), and much less ready to report the other faces even if they were actually the test stimulus (shown by the squares). As for the tilt aftereffect, discrimination is biased *away* from the adapted stimulus. Figure 3C shows that adapting to anti-Adam offers the greatest boost to the event that Adam is reported to a test face (say Henry) that is not Adam, at the *average* face. Reporting Adam falls off if either increasing strengths of Henry *or* anti-Henry are presented. That presenting Henry should decrease the reporting of Adam is obvious, and is commented on in the paper. However, that presenting anti-Henry should decrease the reporting of Adam is less obvious, since, by removing Henry as a competitor, one might have expected Adam to have received an additional boost.

Figure 3B;D shows our factor analysis model of these results. Here, we consider a case with 25 visible units, and 4 factors, one for each face, with generative weights $\mathbf{G} = \{\mathbf{g}^{Adam}, \ldots\}$ governing the input activity associated with full strength versions of each face generated from independent $\mathcal{N}(\mathbf{0}, \mathcal{I})$ distributions. In this representation, morphing is easy, consist-

ing of presenting $\mathbf{u} = \alpha \mathbf{g}^{\text{Adam}} + \boldsymbol{\epsilon}$ where $\alpha$ is the strength and $\boldsymbol{\epsilon}$ is noise (variance $\sigma^2$). The outputs $\mathbf{v} = \mathbf{R}\mathbf{u}$ depend on $\alpha$, the angle between the $\mathbf{g}^i$ and the noise. Next, we need to specify how discrimination is based on the information provided by $\mathbf{v}$. For reasons discussed below, we considered a normalization pool[17,37] for the outputs, treating $(v_i - \min v)^\gamma / \sum (v_j - \min v)^\gamma$ as the probability that face $i$ would be reported, where $\gamma$ is a discrimination parameters. Adaptation to anti-Adam was represented by setting $\bar{\mathbf{u}} = -\mu \mathbf{g}^{\text{Adam}}$, where $\mu$ is the strength of the adapting stimulus.

Figure 3B shows the model of the basic adaptation effect seen in figure 3A. Adapting to $-\mu \mathbf{g}^{\text{Adam}}$ clearly boosts the willingness of the model to report Adam, much as for the subjects. The model is a little more extreme than the average over the subjects. The results for two individual subjects presented in the paper[24] are just as extreme; other subjects may have had softer decision biases. Figure 3D shows the model of figure 3C. The dashed line shows that without the normalization pool, presenting anti-Henry does indeed boost reporting of Adam, when anti-Adam was the adapting stimulus. However, under the above normalization, decreasing $v_i$ boosts the relative strengths of Jim and John (through the minimization in the normalization pool), allowing them to compete, and so reduces the propensity to report Adam (solid line).

# 5 Discussion

We have studied how plasticity associated with adaptation fits with regular unsupervised learning models, in particular factor analysis. It was obvious that there should be a close relationship; this was, however, obscured by aspects of the redundancy reduction models such as the existence of multiple sources of added noise and non-informational constraints. Uniquenesses in factor analysis are exactly the correct noise model for the simple information maximization scheme. We illustrated the model for the case of a simple, linear, model of the tilt aftereffect, and of adaptation in face discrimination. The latter had the interesting wrinkle that the experimental data support something like a normalization pool.[17,37]

Under this current conceptual scheme for adaptation, assumed changes in the input statistics $\mathcal{U}$ are fully compensated for by the factor analysis model (and the linear and Gaussian nature of the model implies that $\bar{\mathbf{u}}$ can be changed without any consequence for the generative or recognition models). The dynamical form of the factor analysis model in equation 6 suggests other possible targets for adaptation. Of particular interest is the possibility that the top-down weights $\mathbf{G}$ and/or the uniquenesses $\mathbf{\Psi}$ might change whilst bottom-up weights $\mathbf{S}$ remain constant. The rationale for this comes from suggestive neurophysiological evidence that bottom-up pathways show delayed plasticity in certain circumstances;[13] and indeed it is exactly what happens in unsupervised learning techniques such as the wake-sleep algorithm.[18,29] Given satisfaction of an eigenvalue condition that the differential equation 6 be stable, it will be interesting to explore the consequences of such changes.

Of course, factor analysis is insufficiently powerful to be an adequate model for cortical unsupervised learning or indeed all aspects of adaptation (as already evident in the limited range of applicability of the model of the tilt aftereffect). However, the ideas about the extraction of higher order statistical structure in the inputs into latent variables, the roles of noise, and the way in equation 6 that predictive coding or explaining away controls cortical representations,[32] survive into sophisticated complex unsupervised learning models,[19] and offer routes for extending the present results.

A paradoxical aspect of adaptation, which neither we nor others have addressed, is the way that the systems that are adapting interact with those to which they send their output. For instance, it would seem unfortunate if all cells in primary visual cortex have to know the light level governing adaptation in order to be able correctly to interpret the information coming bottom-up from the thalamus. In some cases, such as the approximate noise filter $\mathbf{F}$, there are alternative semantics for the adapted neural activity under which this is unnecessary; understanding how this generalizes is a major task for future work.

## Acknowledgements

Funding was from the Gatsby Charitable Foundation. We are most grateful to Odelia Schwartz for discussion and comments.

## Footnotes

*Note that changing the mean $\bar{\mathbf{u}}$ according to the input has no effect on the factor.

## References

[1] Abbott, LF, Varela, JA, Sen, K, & Nelson, SB (1997) Synaptic depression and cortical gain control. *Science* **275**, 220-224.

[2] Atick, JJ (1992) Could information theory provide an ecological theory of sensory processing? *Network: Computation in Neural Systems* **3**, 213-251.

[3] Atick, JJ, & Redlich, AN (1990) Towards a theory of early visual processing. *Neural Computation* **2**, 308-320.

[4] Attneave, F (1954) Some informational aspects of visual perception. *Psychological Review* **61**, 183-193.

[5] Barlow, HB (1961) Possible principles underlying the transformation of sensory messages. In WA Rosenblith, ed., *Sensory Communication*. Cambridge, MA: MIT Press.

[6] Barlow, HB (1969) Pattern recognition and the responses of sensory neurones.,*Annals of the New York Academy of Sciences* **156**, 872-881.

[7] Barlow, HB (1989) Unsupervised learning, *Neural Computation,* **1**, 295-311.

[8] Barlow, H (2001) Redundancy reduction revisited. *Network* **12**,:241-253.

[9] Campbell, FW & Maffei, L (1971) The tilt after-effect: a fresh look. *Vision Research* **11**, 833-40.

[10] Clifford, CWG, Wenderoth, P & Spehar, B. (2000) A functional angle on some after-effects in cortical vision, *Proceedings of the Royal Society of London, Series B* **267**, 1705-1710.

[11] Davis, PJ (1979) *Circulant Matrices.* New York, NY: Wiley.

[12] Dayan, P & Abbott, LF (2001). *Theoretical Neuroscience.* Cambridge, MA: MIT Press.

[13] Diamond, ME, Huang, W & Ebner, FF (1994) Laminar comparison of somatosensory cortical plasticity. *Science* **265**, 1885-1888.

[14] Dong, DW, & Atick, JJ (1995) Temporal decorrelation: A theory of lagged and nonlagged responses in the lateral geniculate nucleus. *Network: Computation in Neural Systems* **6**, 159-178.

[15] Everitt, BS (1984) *An Introduction to Latent Variable Models,* London: Chapman and Hall.

[16] Gibson, JJ & Radner, M (1937) Adaptation, after-effect and contrast in the perception of tilted lines. *Journal of Experimental Psychology* **20**, 453-467.

[17] Heeger, DJ (1992) Normalization of responses in cat striate cortex. *Visual Neuroscience* **9**, 181-198.

[18] Hinton, GE, Dayan, P, Frey, BJ, & Neal, RM (1995) The wake-sleep algorithm for unsupervised neural networks. *Science* **268**, 1158-1160.

[19] Hinton, GE & Sejnowski, TJ (1999) *Unsupervised Learning.* Cambridge, MA: MIT Press.

[20] Jolliffe, IT (1986) *Principal Component Analysis,* New York: Springer.

[21] Jöreskog, KG (1967) Some contributions to maximum likelihood factor analysis, *Psychometrika,* **32**, 443-482.

[22] Jöreskog, KG (1969) A general approach to confirmatory maximum likelihood factor analysis, *Psychometrika,* **34**, 183-202.

[23] Kohonen, T & Oja, E (1976) Fast adaptive formation of orthogonalizing filters and associative memory in recurrent networks of neuron-like elements. *Biological Cybernetics* **21**, 85-95.

[24] Leopold, DA, O'Toole, AJ, Vetter, T & Blanz, V (2001). Prototype-referenced shape encoding revealed by high-level aftereffects. *Nature Neuroscience* **4**, :89-94.

[25] Li, Z & Atick, JJ (1994a) Efficient stereo coding in the multiscale representation. *Network: Computation in Neural Systems* **5**, 157-174.

[26] Linsker, R (1988) Self-organization in a perceptual network, *Computer,* **21**, 105-128.

[27] Maguire G, Hamasaki DI (1994) The retinal dopamine network alters the adaptational properties of retinal ganglion cells in the cat.*Journal of Neurophysiology,* **72**, 730-741.

[28] McCormick, DA (1990) Membrane properties and neurotransmitter actions. In GM Shepherd, ed., *The Synaptic Organization of the Brain.* New York: Oxford University Press.

[29] Neal, RM & Dayan, P (1997). Factor Analysis using delta-rule wake-sleep learning. *Neural Computation,* **9**, 1781-1803.

[30] Neisser, U (1967) *Cognitive Psychology.* New York: Appleton-Century-Crofts.

[31] Olshausen, BA, & Field, DJ (1996) Emergence of simple-cell receptive field properties by learning a sparse code for natural images. *Nature* **381**, 607-609.

[32] Rao, RPN, & Ballard, DH (1997) Dynamic model of visual recognition predicts neural response properties in the visual cortex. *Neural Computation* **9**, 721-763.

[33] Regan, D & Beverley, KI (1985) Postadaptation orientation discrimination. *JOSA A,* **2**, 147-155.

[34] Roweis, S & Ghahramani, Z (1999) A unifying review of linear gaussian models. *Neural Computation* **11**, 305-345.

[35] Rubin, DB & Thayer, DT (1982) EM algorithms for ML factor analysis, *Psychometrika,* **47**, 69-76.

[36] Ruderman DL & Bialek W (1994) Statistics of natural images: Scaling in the woods. *Physical Review Letters* **73**, 814-817.

[37] Schwartz, O & Simoncelli, EP (2001) Natural signal statistics and sensory gain control. *Nature Neuroscience* **4**, 819-825.

[38] Shapley, R & Enroth-Cugell, C (1984) Visual adaptation and retinal gain control. *Progress in Retinal Research* **3**, 263-346.

[39] Tipping, ME & Bishop, CM (1999) Mixtures of probabilistic principal component analyzers. *Neural Computation* **11**, 443-482.

[40] van Hateren, JH (1992) A theory of maximizing sensory information. *Biological Cybernetics* **68**, 23-29.

[41] Wainwright, MJ (1999) Visual adaptation as optimal information transmission. *Vision Research* **39**, 3960-3974.

[42] Weiler R & Wagner HJ (1984) Light-dependent change of cone-horizontal cell interactions in carp retina. *Brain Resesarch* **298**, 1-9.
